# Vicinal Risk Minimization

**Olivier Chapelle, Jason Weston\*, Léon Bottou and Vladimir Vapnik**
AT&T Research Labs, 100 Schultz drive, Red Bank, NJ, USA
\* Barnhill BioInformatics.com, Savannah, GA, USA.
{*chapelle,weston,leonb,vlad*}*@research.att.com*

## Abstract

The Vicinal Risk Minimization principle establishes a bridge between generative models and methods derived from the Structural Risk Minimization Principle such as Support Vector Machines or Statistical Regularization. We explain how VRM provides a framework which integrates a number of existing algorithms, such as Parzen windows, Support Vector Machines, Ridge Regression, Constrained Logistic Classifiers and Tangent-Prop. We then show how the approach implies new algorithms for solving problems usually associated with generative models. New algorithms are described for dealing with pattern recognition problems with very different pattern distributions and dealing with unlabeled data. Preliminary empirical results are presented.

## 1 Introduction

Structural Risk Minimisation (SRM) in a learning system can be achieved using constraints on the parameter vectors, using regularization terms in the cost function, or using Support Vector Machines (SVM). All these principles have lead to well established learning algorithms.

It is often said, however, that some problems are best addressed by generative models. The first problem is of missing data. We may for instance have a few labeled patterns and a large number of unlabeled patterns. Intuition suggests that these unlabeled patterns carry useful information. The second problem is of discriminating classes with very different pattern distributions. This situation arises naturally in anomaly detection systems. This also occurs often in recognition systems that reject invalid patterns by defining a garbage class for grouping all ambiguous or unrecognizable cases. Although there are successful non-generative approaches (Schuurmans and Southey, 2000) (Drucker, Wu and Vapnik, 1999), the generative framework is undeniably appealing. Recent results (Jaakkola, Meila and Jebara, 2000) even define generative models that contain SVM as special cases.

This paper discusses the Vicinal Risk Minimization (VRM) principle, summarily introduced in (Vapnik, 1999). This principle was independently hinted at by Tong and Koller (Tong and Koller, 2000) with a useful generative interpretation. In particular, they proved that SVM are a limiting case of their Restricted Bayesian Classifiers. We extend Tong's and Koller's result by showing that VRM subsumes several well known techniques such as Ridge Regression (Hoerl and Kennard, 1970), Constrained Logistic Classifier, or Tangent Prop (Simard et al., 1992). We then go on to show how VRM naturally leads to simple algo-

rithms that can deal with problems for which one would have formally considered purely generative models. We provide algorithms and preliminary empirical results for dealing with unlabeled data or recognizing classes with very different pattern distributions.

## 2 Vicinal Risk Minimization

The learning problem can be formulated as the search of the function $f \in \mathcal{F}$ that minimizes the expectation of a given loss $\ell(f(\mathbf{x}), y)$.

$$R(f) = \int \ell(f(\mathbf{x}), y) \, dP(\mathbf{x}, y) \tag{1}$$

In the classification framework, $y$ takes values $\pm 1$ and $\ell(f(\mathbf{x}), y)$ is a step function such as $1 - \text{Sign}(yf(\mathbf{x}))$, whereas in the regression framework, $y$ is a real number and commonly $\ell(f(\mathbf{x}), y)$ is the mean squared error $(f(\mathbf{x}) - y)^2$.

The expectation (1) cannot be computed since the distribution $P(\mathbf{x}, y)$ is unknown. However, given a training set $\{(\mathbf{x}_i, y_i)\}_{1 \leq i \leq n}$, it is common to minimize instead the empirical risk:

$$R_{emp}(f) = \frac{1}{n} \sum_{i=1}^{n} \ell(f(\mathbf{x}_i), y_i)$$

Empirical Risk Minimization (ERM) is therefore equivalent to minimizing the expectation of the loss function with respect to an empirical distribution $P_{emp}(\mathbf{x}, y)$ formed by assembling delta functions located on each example:

$$dP_{emp}(\mathbf{x}, y) = \frac{1}{n} \sum_{i=1}^{n} \delta_{\mathbf{x}_i}(\mathbf{x}) \delta_{y_i}(y) \tag{2}$$

It is quite natural to consider improved density estimates by replacing the delta functions $\delta_{\mathbf{x}_i}(\mathbf{x})$ by some estimate of the density in the *vicinity* of the point $\mathbf{x}_i$, $P_{\mathbf{x}_i}(\mathbf{x})$.

$$dP_{est}(\mathbf{x}, y) = \frac{1}{n} \sum_{i=1}^{n} dP_{\mathbf{x}_i}(\mathbf{x}) \delta_{y_i}(y) \tag{3}$$

We can define in this way the *vicinal risk* of a function as:

$$R_{vic}(f) = \int \ell(f(\mathbf{x}), y) \, dP_{est}(\mathbf{x}, y) = \frac{1}{n} \sum_{i=1}^{n} \int \ell(f(\mathbf{x}), y_i) dP_{\mathbf{x}_i}(\mathbf{x}) \tag{4}$$

The Vicinal Risk Minimization principle consists of estimating $\text{argmin}_{f \in \mathcal{F}} R(f)$ by the function which minimizes the vicinal risk (4). In general, one can construct the VRM functional using any estimate $dP_{est}(\mathbf{x}, y)$ of the density $dP(\mathbf{x}, y)$, instead of restricting our choices to pointwise kernel estimates.

Spherical gaussian kernel functions $\mathcal{N}_\sigma(\mathbf{x} - \mathbf{x}_i)$ are otherwise an obvious choice for the local density estimate $dP_{\mathbf{x}_i}(x)$. The corresponding density estimate $dP_{est}$ is a Parzen windows estimate. The parameter $\sigma$ controls the scale of the density estimate. The extreme case $\sigma = 0$ leads to the estimation of the density by delta functions and therefore leads to ERM. This must be distinguished from the case $\sigma \to 0$ because the limit is taken *after* the minimization of the integral, leading to different results as shown in the next section.

The theoretical analysis of ERM (Vapnik, 1999) shows that the crucial factor is the capacity of the class $\mathcal{F}$ of functions. Large classes entail the risk of overfitting, whereas small classes entail the risk of underfitting. Two factors however are responsible for generalization of VRM, namely the quality of the estimate $dP_{est}$ and the size of the class $\mathcal{F}$ of functions.

If $dP_{est}$ is a poor approximation to $P$ then VRM can still perform well if $\mathcal{F}$ has suitably small capacity. ERM indeed uses a very naive estimate of $dP$ and yet can provide good results. On the other hand, if $\mathcal{F}$ is not chosen with suitably small capacity then VRM can still perform well if the estimate $dP_{est}$ is a good approximation to $dP$. One can even take the set of all possible functions (whose capacity is obviously infinite) and still find a good solution if the estimate $dP_{est}$ is close enough to $dP$ with an adequate metric. For example, if $dP_{est}$ is a Parzen window density estimate, then the Vicinal Risk minimizer is the Parzen window classifier. This latter property contrasts nicely with the ERM principle whose results strongly depend on the choice of the class of functions. Although we do not have a full theoretical understanding of VRM at this time, we expect considerable differences in the theoretical analysis of ERM and VRM.

## 3  Special Cases

We now discuss the relationship of VRM to existing methods. There are obvious links between VRM and Parzen windows or Nearest Neighbour when the set of functions $\mathcal{F}$ is unconstrained. Furthermore, many existing algorithms can be viewed as special cases of VRM for different choices of $\mathcal{F}$ and $dP_{est}$.

**a) VRM Regression and Ridge Regression** — Consider the case of VRM for regression with spherical Parzen windows (using gaussian kernel) with standard deviation $\sigma$ and with a family $\mathcal{F}$ of linear functions $f_{\mathbf{w},b}(\mathbf{x}) = \mathbf{w} \cdot \mathbf{x} + b$. We can write the vicinal risk as:

$$
\begin{aligned}
R_{vic}(f) &= \frac{1}{n} \sum_{i=1}^{n} \int (f(\mathbf{x}) - y_i)^2 dP_{\mathbf{x}_i}(\mathbf{x}) \\
&= \frac{1}{n} \sum_{i=1}^{n} \int (f(\mathbf{x}_i + \boldsymbol{\epsilon}) - y_i)^2 d\mathcal{N}_\sigma(\boldsymbol{\epsilon}) \\
&= \frac{1}{n} \sum_{i=1}^{n} \int (f(\mathbf{x}_i) - y_i + \mathbf{w} \cdot \boldsymbol{\epsilon})^2 \, d\mathcal{N}_\sigma(\boldsymbol{\epsilon}) \\
&= \frac{1}{n} \sum_{i=1}^{n} (f(\mathbf{x}_i) - y_i)^2 \; + \; \sigma^2 \|\mathbf{w}\|^2
\end{aligned}
$$

The resulting expression is the empirical risk augmented by a regularization term. The particular cost function above is known as the Ridge Regression cost function (Hoerl and Kennard, 1970).

This result can be extended to the case of non linear functions $f$ by performing a Taylor expansion of $f(x_i + \boldsymbol{\epsilon})$. The corresponding regularization term then combines successive derivatives of function $f$. Useful mathematical arguments can be found in (Leen, 1995).

**b) VRM and Invariant Learning** — Generating synthetic examples is a simple way to incorporate selected invariances in a learning system. For instance, we can augment a optical character recognition database by applying applying translations or rotations to the initial examples. In the limit, this is equivalent to replacing each initial example by a distribution whose shape represents the desired invariances. This formulation naturally leads to a special case of VRM in which the local density estimates $P_{x_i}(x)$ are elongated in the direction of invariance.

Tangent-Prop (Simard et al., 1992) is a more sophisticated way to incorporate invariances by adding an adequate regularization term to the cost function. Tangent-Prop has been formally proved to be equivalent to generating synthetic examples with infinitesimal deformations (Leen, 1995). This analysis makes Tangent-Prop a special case of VRM. The local

density estimate $P_{\mathbf{x}_i}$ is simply formed by Gaussian kernels with a covariance matrix whose eigenvectors describe the tangent direction to the invariant manifold. The eigenvalues then represent the respective strengths of the selected invariances.

The tangent covariance matrix used in the SVM context by (Schölkopf et al., 1998) specifies invariances globally. It can also been seen as a special case of VRM.

**c) VRM Classifier and Constrained Logistic Classifier** — Consider the case of VRM for classification with spherical Parzen windows with standard deviation $\sigma$ and with a family $\mathcal{F}$ of linear functions $f_{\mathbf{w},b}(\mathbf{x}) = \mathbf{w} \cdot \mathbf{x} + b$. We can assume without loss of generality that $\|\mathbf{w}\| = 1$. We can write the vicinal risk as:

$$
\begin{aligned}
R_{vic}(\mathbf{w},b) &= \frac{1}{n}\sum_{i=1}^{n}\int -y_i \,\mathrm{Sign}(b + \mathbf{w}\cdot\mathbf{x})\, dP_{\mathbf{x}_i}(\mathbf{x}) \\
&= \frac{1}{n}\sum_{i=1}^{n} -y_i \int \mathrm{Sign}(b + \mathbf{w}\cdot\mathbf{x}_i + \mathbf{w}\cdot\boldsymbol{\epsilon})\, d\mathcal{N}_\sigma(\boldsymbol{\epsilon})
\end{aligned}
$$

We can decompose $\boldsymbol{\epsilon} = \mathbf{w}\epsilon_w + \boldsymbol{\epsilon}_\perp$ where $\mathbf{w}\epsilon_w$ represents its component parallel to $\mathbf{w}$ and $\boldsymbol{\epsilon}_\perp$ represents its orthogonal component. Since $\|\mathbf{w}\| = 1$, we have $\mathbf{w}\cdot\boldsymbol{\epsilon} = \epsilon_w$. After integrating over $\boldsymbol{\epsilon}_\perp$ we are left with the following expression:

$$
R_{vic}(\mathbf{w},b) = \frac{1}{n}\sum_{i=1}^{n} -y_i \int \mathrm{Sign}(b + \mathbf{w}\cdot\mathbf{x}_i + \epsilon_w)\, d\mathcal{N}_\sigma(\epsilon_w)
$$

The latter integral can be seen as the convolution of the Gaussian $\mathcal{N}_\sigma(x)$ with the step function $\mathrm{Sign}(x)$, which is a sigmoid shaped function with asymptotes at $\pm 1$. Using notation $\varphi(x) = 2\,\mathrm{erf}(x) - 1$, we can write:

$$
R_{vic}(\mathbf{w},b) = \frac{1}{n}\sum_{i=1}^{n} -y_i\, \varphi\left(\frac{\mathbf{w}\cdot\mathbf{x}_i + b}{\sigma}\right)
$$

By rescaling $\mathbf{w}$ and $b$ by a factor $1/\sigma$, we can write the following equivalent formulation of the VRM:

$$
\left\{
\begin{aligned}
& \underset{\mathbf{w},b}{\mathrm{Arg\,Min}} \quad -\frac{1}{n}\sum_{i=1}^{n} y_i\, \varphi(\mathbf{w}\cdot\mathbf{x}_i + b) \\
& \text{with constraint } \|\mathbf{w}\| = 1/\sigma
\end{aligned}
\right.
\tag{5}
$$

Except for the minor shape difference between sigmoid functions, the above formulation describes a Logistic Classifier with a constraint on the weights. This formulation is also very close to using a single artificial neuron with a sigmoid transfer function and a weight decay.

The above proof illustrates a general identity. Transforming the empirical probability estimate (2) by convolving it with a kernel function is equivalent to transforming the loss function $\ell(f(x), y)$ by convolving it with the same kernel function. This is summarized in the following equality, where $\star$ represents the convolution operator.

$$
\int \ell(f(x), y)\, [\mathcal{N}_\sigma(\cdot) \star dP_{emp}(\cdot, y)](\mathbf{x}) = \int [\ell(f(\cdot), y) \star \mathcal{N}_\sigma(\cdot)](\mathbf{x})\, dP_{emp}(\mathbf{x}, y)
$$

**d) VRM Classifier and SVM (Tong and Koller, 2000)** — Consider *again* the case of VRM for classification with spherical Parzen windows with standard deviation $\sigma$ and with a family $\mathcal{F}$ of linear functions $f_{\mathbf{w},b}(\mathbf{x}) = \mathbf{w} \cdot \mathbf{x} + b$. The resulting algorithm is in fact a Restricted Bayesian Classifier (Tong and Koller, 2000). Assuming that the examples are

separable, Tong and Koller have shown that the resulting decision boundary tends towards the hard margin SVM decision boundary when $\sigma$ tends towards zero.

The proof is based on the following observation: when $\sigma \rightarrow 0$, the vicinal risk (4) is dominated by the terms corresponding to the examples whose distance to the decision boundary is minimal. These examples in fact are the support vectors. On the other hand, choosing $\sigma > 0$ generates a decision boundary which depends on all the examples. The contribution of each example decreases exponentially when its distance to the decision boundary increases. This is only slightly different from a soft margin SVM whose boundary relies on support vectors that can be more distant than those selected by hard margin SVM. The difference here is just in the cost functions (sigmoid compared to linear loss).

e) **SVM and Constrained Logistic Classifiers** — The two previous paragraphs show that the same particular case of VRM is (a) equivalent to a Logistic Classifier with a constraint on the weights, and (b) tends towards the SVM classifier when $\sigma \rightarrow 0$ and when the examples are separable. As a consequence, we can state that the Logistic Classifier decision boundary tends towards the SVM decision boundary when we relax the constraint on the weights.

In practice we can find the SVM solution with a Logistic Classifier by simply using an iterative weight update algorithm such as gradient descent, choosing small initial weights, and letting the norm of the weights grow slowly while the iterative algorithm is running. Although this algorithm is not exact, it is fast and efficient. This is in fact similar to what is usually done with back-propagation neural networks (LeCun et al., 1998). The same algorithm can be used for the VRM. In that context *early stopping* is similar to choosing the optimal $\sigma$ using cross-validation.

# 4 New Algorithms and Results

## 4.1 Adaptive Kernel Widths

It is known in density estimation theory that the quality of the density estimate can be improved using variable kernel widths (Breiman, Meisel and Purcell, 1977). In regions of the space where there is little data, it is safer to have a smooth estimate of the density, whereas in the regions of the space there is more data one wants to be as accurate as possible via sharper kernel estimates. The VRM principle can take advantage of these improved density estimates for other problem domains. We consider here the following density estimate:

$$dP_{est}(\mathbf{x}, y) = \frac{1}{n} \sum_i \delta_{y_i}(y) \, \mathcal{N}_{\sigma_i}(\mathbf{x} - \mathbf{x}_i) \, d\mathbf{x}$$

where the specific kernel width $\sigma_i$ for each training example $\mathbf{x}_i$ is computed from the training set.

a) **Wisconsin Breast Cancer** — We made a first test of the method on the Wisconsin breast cancer dataset[1] which contains 589 examples on 30 dimensions. We compared VRM using the set of linear classifiers with various underlying density estimates. The minimization was achieved using gradient descent on the vicinal risk. All hyperparameters were determined using cross-validation. The following table reports results averaged on 100 runs.

| Training Set | Hard SVM | Soft SVM Best $C$ | VRM Best fixed $\sigma$ | VRM Adaptive $\sigma_i$ |
|---|---|---|---|---|
| 10 | 11.3% | 11.1% | 10.8% | **9.6%** |
| 20 | 8.3% | 7.5% | 6.9% | **6.6%** |
| 40 | 6.3% | 5.5% | 5.2% | **4.8%** |
| 80 | 5.4% | 4.0% | 3.9% | **3.7%** |

The adaptive kernel width $\sigma_i$ were computed by multiplying a global factor by the average distance of the five closest training examples. The best global factor is determined by cross-validation. These results suggest that VRM with adaptive kernel widths can outperform state of the art classifiers on small training sets.

**b) MNIST "1" versus other digits** — A second test was performed using the MNIST handwritten digits[2]. We considered the sub-problem of recognizing the ones versus all other digits. The testing set contains 10000 digits (5000 ones and 5000 non-ones). Two training set sizes were considered with 250 or 500 ones and an equal number of non-ones. Computations were achieved using the algorithm suggested in section (3.e). We simply trained a single linear unit with a sigmoid transfer function using stochastic gradient updates. This is appropriate for implementing an approximate VRM with a single kernel width. Adaptive kernel widths are implemented by simply changing the slope of the sigmoid for each example. For each example $\mathbf{x}_i$, the kernel width $\sigma_i$ is computed from the training set using the 5/1000th quantile of the distances of all other examples to example $\mathbf{x}_i$. The sigmoid slopes are then computed by renormalizing the $\sigma_i$ in order to make their mean equal to 1. Early stopping was achieved with cross-validation.

| Training Set | Hard SVM | VRM Fixed slope | VRM Adaptive slope |
|---|---|---|---|
| 250+250 | 3.34% | 2.79% | **2.54%** |
| 500+500 | 3.11% | 2.47% | **2.27%** |
| 1000+1000 | 2.94% | 2.08% | **1.96%** |

The statistical signifiance of these results can be asserted with very high probability by comparing the list of errors performed by each system (Bottou and Vapnik, 1992). Again these results suggest that VRM with adaptive kernel widths can be very useful with small training sets.

## 4.2  Unlabeled Data

In some applications unlabeled data is in abundance whereas labeled data is not. The use of unlabeled data falls into the framework of VRM by simply making the same vicinal loss for unlabeled points. Given $m$ unlabeled points $x_1^*, \ldots, x_m^*$, one obtains the following formulation:

$$R_{vic}(f) = \frac{1}{n} \sum_{i=1}^{n} \int \ell(f(\mathbf{x}), y_i) dP_{\mathbf{x}_i}(\mathbf{x}) + \frac{1}{m} \sum_{i=1}^{m} \int \ell(f(\mathbf{x}), f(\mathbf{x}_i^*)) dP_{\mathbf{x}_i^*}(\mathbf{x}).$$

To give an example of the usefulness of our approach consider the following example. Two normal distributions on the real line $\mathcal{N}(-1.6, 1)$ and $\mathcal{N}(1.6, 1)$ model the patterns of two classes with equal probability; 20 labeled points and 100 unlabeled points are drawn. The following table compares the true generalization error of VRM with gaussian kernels and linear functions. Results are averaged over 100 runs. Two different kernel widths $\sigma_L$ and $\sigma_U$ were used for kernels associated with labeled or unlabeled examples. Best kernel widths were obtained by cross-validation. We also studied the case $\sigma_L \to 0$ in order to provide a result equivalent to a plain SVM.

| $\sigma_L$ | $\sigma_U$ | Labeled | Labeled+Unlabeled |
|---|---|---|---|
| $\sigma_L \to 0$ | Best $\sigma_U$ | 6.5% | 5.6% |
| Best $\sigma_L$ | Best $\sigma_U$ | 5.0% | **4.3%** |

Note that when both $\sigma_L$ and $\sigma_U$ tend to zero, this algorithm reverts to a transduction algorithm due to Vapnik which was previously solved by the more difficult optimization procedure of integer programming (Bennet and Demiriz, 1999).

## 5 Conclusion

In conclusion, the Vicinal Risk Minimization VRM principle provides a useful bridge between generative models and SRM methods such as SVM or Statistic Regularization. Several well known algorithms are in fact special cases of VRM. The VRM principle also suggests new algorithms. In this paper we proposed algorithms for dealing with unlabeled data and recognizing classes with very different pattern distributions, obtaining initial promising results. We hope that this approach can lead to further understanding of existing methods and also to suggest new ones.

## Footnotes

[1] http://horn.first.gmd.de/~raetsch/data/breast-cancer.

[2]http://www.research.att.com/~yann/ocr/index.html

## References

Bennet, K. and Demiriz, A. (1999). Semi-supervised support vector machines. In *Advances in Neural Information Processing Systems 11*, pages 368–374. MIT Press.

Bottou, L. and Vapnik, V. N. (1992). Local learning algorithms, appendix on confidence intervals. *Neural Computation*, 4(6):888–900.

Breiman, L., Meisel, W., and Purcell, E. (1977). Variable kernel estimates of multivariate densities. *Technometrics*, 19:135–144.

Drucker, H., Wu, D., and Vapnik, V. (1999). Support vector machines for spam categorization. *Neural Networks*, 10:1048–1054.

Hoerl, A. and Kennard, R. W. (1970). Ridge regression: Biased estimation for nonorthogonal problems. *Technometrics*, 12(1):55–67.

Jaakkola, T., Meila, M., and Jebara, T. (2000). Maximum entropy discrimination. In *Advances in Neural Information Processing Systems 12*. MIT Press.

LeCun, Y., Bottou, L., Orr, G., and Muller, K. (1998). Efficient backprop. In Orr, G. and K., M., editors, *Neural Networks: Tricks of the Trade*. Springer.

Leen, T. K. (1995). Invariance and regularization in learning. In *Advances in Neural Information Processing Systems 7*. MIT Press.

Schölkopf, B., Simard, P., Smola, A., Vapnik, V. (1998). Prior knowledge in support vector kernels. In *Advances in Neural Information Processing Systems 10*. MIT Press.

Schuurmans, D. and Southey, F. (2000). An adaptive regularization criterion for supervised learning. In *Proceedings of the Seventeenth International Conference on Machine Learning (ICML-2000)*.

Simard, P., Victorri, B., Le Cun, Y., and Denker, J. (1992). Tangent prop: a formalism for specifying selected invariances in adaptive networks. In *Advances in Neural Information Processing Systems 4*, Denver, CO. Morgan Kaufman.

Tong, S. and Koller, D. (2000). Restricted bayes optimal classifiers. *Proceedings of the 17th National Conference on Artificial Intelligence (AAAI)*.

Vapnik, V. (1999). *The Nature of Statistical Learning Theory (Second Edition)*. Springer Verlag, New York.
